# Wavelet based multi-scale shape features on *arbitrary* surfaces for cortical thickness discrimination

**Won Hwa Kim**[†¶*] **Deepti Pachauri**[†] **Charles Hatt**[‡]

**Moo K. Chung**[§] **Sterling C. Johnson**[*¶] **Vikas Singh**[§†*¶]

[†]Dept. of Computer Sciences, University of Wisconsin, Madison, WI
[§]Dept. of Biostatistics & Med. Informatics, University of Wisconsin, Madison, WI
[‡]Dept. of Biomedical Engineering, University of Wisconsin, Madison, WI
[¶]Wisconsin Alzheimer's Disease Research Center, University of Wisconsin, Madison, WI
[*]GRECC, William S. Middleton VA Hospital, Madison, WI

{wonhwa, pachauri}@cs.wisc.edu {hatt, mkchung}@wisc.edu
scj@medicine.wisc.edu vsingh@biostat.wisc.edu

## Abstract

Hypothesis testing on signals defined on surfaces (such as the cortical surface) is a fundamental component of a variety of studies in Neuroscience. The goal here is to identify regions that exhibit changes as a function of the clinical condition under study. As the clinical questions of interest move towards identifying *very* early signs of diseases, the corresponding statistical differences at the group level invariably become weaker and increasingly hard to identify. Indeed, after a multiple comparisons correction is adopted (to account for correlated statistical tests over all surface points), very few regions may survive. In contrast to hypothesis tests on point-wise measurements, in this paper, we make the case for performing statistical analysis on multi-scale shape descriptors that characterize the local topological context of the signal around each surface vertex. Our descriptors are based on recent results from harmonic analysis, that show how wavelet theory extends to non-Euclidean settings (i.e., irregular weighted graphs). We provide strong evidence that these descriptors successfully pick up group-wise differences, where traditional methods either fail or yield unsatisfactory results. Other than this primary application, we show how the framework allows performing cortical surface smoothing in the native space without mappint to a unit sphere.

## 1 Introduction

Cortical thickness measures the distance between the *outer* and *inner* cortical surfaces (see Fig. 1). It is an important biomarker implicated in brain development and disorders [3]. Since 2011, more than 1000 articles (from a search on Google Scholar and/or Pubmed) tie cortical thickness to conditions ranging from Alzheimer's disease (AD), to Schizophrenia and Traumatic Brain injury (TBI) [9, 14, 13]. Many of these results show how cortical thickness also correlates with brain growth (and atrophy) during adolescence (and aging) respectively [22, 20, 7]. Given that brain function and pathology manifest strongly as changes in the cortical thickness, the statistical analysis of such data (to find group level differences in clinically disparate populations) plays a central role in structural neuroimaging studies.

In typical cortical thickness studies, magnetic resonance images (MRI) are acquired for two populations: clinical and normal. A sequence of image processing steps are performed to segment the cortical surfaces and establish vertex-to-vertex correspondence across surface meshes [15]. Then, a group-level analysis is performed at *each* vertex. That is, we can ask if there are statistically significant differences in the signal between the two groups. Since there are multiple correlated statistical

tests over all voxels, a Bonferroni type multiple comparisons correction is required [4]. If many vertices survive the correction (i.e., differences are strong enough), the analysis will reveal a set of *discriminative cortical surface regions*, which may be positively or negatively correlated with the clinical condition of interest. This procedure is well understood and routinely used in practice.

In the last five years, a significant majority of research has shifted towards investigations focused on the pre-clinical stages of diseases [16, 23, 17]. For instance, we may be interested in identifying *early* signs of dementia by analyzing cortical surfaces (e.g., by comparing subjects that carry a certain gene versus those who do not). In this regime, the differences are weaker, and the cortical differences may be too subtle to be detected. In a statistically under-powered cortical thickness analysis, few vertices may survive the multiple comparisons correction. Another aspect that makes this task challenging is that the cortical thickness data (obtained from state of the art tools) is still inherently noisy. The standard approach for filtering cortical surface noise is to adopt an appropriate parameterization to model the signal followed by a diffusion-type smoothing [6]. The primary difficulty is that most (if not all) widely used parameterizations operate in a spherical coordinate system using spherical harmonic (SPHARM) basis functions [6]. As a result, one must

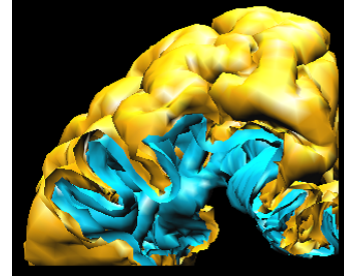

Figure 1: Cortical thickness illustration: the outer cortical surface (in yellow) and the inner cortical surface (in blue). The distance between the two surfaces is the cortical thickness.

first project the signal on the surface to a unit sphere. This "ballooning" process introduces serious metric distortions. Second, SPHARM parameterization usually suffers from *ringing* artifacts (i.e., Gibbs phenomena) when used to fit rapidly changing localized cortical measurements [10]. Third, SPHARM uses *global* basis functions which typically requires a large number of terms in the expansion to model cortical surface signals to high fidelity. Subsequently, even if the globally-based coefficients exhibit statistical differences, interpreting which brain regions contribute to these variations is difficult. As a result, the coefficients of the model cannot be used directly in localizing variations in the cortical signal.

This paper is motivated by the simple observation that statistical inference on surface based signals should be based not on a single scalar measurement but on multivariate descriptors that characterize the *topologically localized context* around each point sample. This view insures against signal noise at individual vertices, and should offer the tools to meaningfully compare the behavior of the signal at multiple resolutions of the topological feature, across multiple subjects. The ability to perform the analysis in a multi-resolution manner, it seems, is addressable if one makes use of Wavelets based methods (e.g., scaleograms [19]). Unfortunately, the non-regular structure of the topology makes this problematic. In our neuroimaging application, samples are not drawn on a regular grid, instead governed entirely by the underlying cortical surface mesh of the participant. To get around this difficulty, we make use of some recent results from the harmonic analysis literature [8] – which suggests how wavelet analysis can be extended to arbitrary weighted graphs with irregular topology. We show how these ideas can be used to derive a wavelet multi-scale descriptor for statistical analysis of signals defined on surfaces. This framework yields rather surprising improvements in discrimination power and promises immediate benefits for structural neuroimaging studies.

**Contributions.** We derive wavelet based multi-scale representations of surface based signals. Our representation has varying levels of local support, and as a result can characterize the local context around a vertex to varying levels of granularity. We show how this facilitates statistical analysis of signals defined on arbitrary topologies (instead of the lattice setup used in image processing).

(i) We show how the new model significantly extends the operating range of analysis of cortical surface signals (such as cortical thickness). At a pre-specified significance level, we can detect a much stronger signal showing group differences that are barely detectable using existing approaches. This is the main experimental result of this paper.

(ii) We illustrate how the procedure of smoothing of cortical surfaces (and shapes) can completely bypass the mapping on to a sphere, since smoothing can now be performed in the native space.

## 2 A Brief Review of Wavelets in Signal Processing

Recall that the celebrated Fourier series representation of a periodic function is expressed via a superposition of sines and cosines, which is widely used in signal processing for representing a signal in the frequency domain and obtaining meaningful information from it. Wavelets are conceptually similar to the Fourier series transform, in that they can be used to extract information from many different kinds of data, however unlike the Fourier transform which is localized in frequency only, wavelets can be localized in both time and frequency [12] and extend frequency analysis to the notion of scale. The construction of wavelets is defined by a wavelet function $\psi$ (called an analyzing wavelet or a mother wavelet) and a scaling function $\phi$. Here, $\psi$ serves as a band-pass filter and $\phi$ operates as a low-pass filter covering the low frequency components of the signal which cannot be tackled by the band-pass filters. When the band-pass filter is transformed back by the inverse transform and translated, it becomes a localized oscillating function with finite duration, providing very compact (local) support in the original domain [21]. This indicates that points in the original domain which are far apart have negligable impact on one another. Note the contrast with Fourier series representation of a short pulse which suffers from issues due to nonlocal support of $\sin(\cdot)$ with infinite duration.

Formally, the wavelet function $\psi$ on $x$ is a function of two parameters, the scale and translation parameters, $s$ and $a$

$$\psi_{s,a}(x) = \frac{1}{a}\psi(\frac{x-a}{s}) \qquad (1)$$

Varying scales control the *dilation* of the wavelet, and together with a *translation* parameter, constitute the key building blocks for approximating a signal using a wavelet expansion. The function $\psi_{s,a}(x)$ forms a basis for the signal and can be used with other bases at different scales to decompose a signal, similar to Fourier transform. The wavelet transform of a signal $f(x)$ is defined as the inner product of the wavelet and signal and can be represented as

$$W_f(s,a) = \langle f, \psi \rangle = \frac{1}{a}\int f(x)\psi^*(\frac{x-a}{s})\mathrm{d}x \qquad (2)$$

where $W_f(s,a)$ is the wavelet coefficient at scale $s$ and at location $a$. The function $\psi^*$ represents the complex conjugate of $\psi$. Such a transform is invertible, that is

$$f(x) = \frac{1}{C_\psi}\iint W_f(s,a)\psi_{s,a}(x)\mathrm{d}a\,\mathrm{d}s \qquad (3)$$

where $C_\psi = \int \frac{|\Psi(j\omega)|^2}{|\omega|}\mathrm{d}\omega$ is called the *admissibility condition constant*, and $\Psi$ is the Fourier transform of the wavelet [21], and the $\omega$ is the domain of frequency.

As mentioned earlier, the scale parameter $s$ controls the dilation of the basis and can be used to produce both short and long basis functions. While short basis functions correspond to high frequency components and are useful to isolate signal discontinuities, longer basis functions corresponding to lower frequencies, are also required to to obtain detailed frequency analysis. Indeed, wavelets transforms have an infinite set of possible basis functions, unlike the single set of basis functions (sine and cosine) in the Fourier transform. Before concluding this section, we note that while wavelets based analysis for image processing is a mature field, most of these results are not directly applicable to non-uniform topologies such as those encountered in shape meshes and surfaces in Fig. 1.

## 3 Defining Wavelets on Arbitrary Graphs

Note that the topology of a brain surface is naturally modeled as a weighted graph. However, the application of wavelets to this setting is not straightforward, as wavelets have traditionally been limited to the Euclidean space setting. Extending the notion of wavelets to a non-Euclidean setting, particularly to weighted graphs, requires deriving a multi-scale representation of a function defined on the vertices. The first bottleneck here is to come up with analogs of *dilation* and *translation* on the graph. To address this problem, in [8], the authors introduce *Diffusion Wavelets* on manifolds. The basic idea is related to some well known results from machine learning, especially the eigenmaps framework by Belkin and Niyogi [1]. It also has a strong relationship with random walks on a weighted graph. Briefly, a graph $G = (V, E, w)$ with vertex set $V$, edge set $E$ and symmetric edge

weights $w$ has an associated random walk $R$. The walk $R$, when represented as a matrix, is conjugate to a self adjoint matrix $T$, which can be interpreted as an operator associated with a diffusion process, explaining how the random walk propagates from one node to another. Higher powers of $T$ (given as $T^t$) induce a dilation (or scaling) process on the function to which it is applied, and describes the behavior of the diffusion at varying time scales ($t$). This is equivalent to iteratively performing a random walk for a certain number of steps and collecting together random walks into representatives [8]. Note that the orthonormalization of the columns of $T$ induces the effect of "compression", and corresponds to downsampling in the function space [5]. In fact, the powers of $T$ are low rank (since the spectrum of $T$ decays), and this ties back to the compressibility behavior of classical wavelets used in image processing applications (e.g., JPEG standard). In this way, the formalization in [8] obtains all wavelet-specific properties including dilations, translations, and downsampling.

## 3.1 Constructing Wavelet Multiscale Descriptors (WMD)

Very recently, [11] showed that while the orthonormalization above is useful for iteratively obtaining compression (i.e., coarser subspaces), it complicates the construction of the transform and only provides limited control on scale selection. These issues are critical in practice, especially when adopting this framework for analysis of surface meshes with $\sim 200,000$ vertices with a wide spectrum of frequencies (which can benefit from finer control over scale). The solution proposed in [11] discards repeated application of the diffusion operator $T$, and instead relies on the graph Laplacian to derive a spectral graph wavelet transform (SGWT). To do this, [11] uses a form of the wavelet operator in the Fourier domain, and generalizes it to graphs. Particularly, SGWT takes the Fourier transform of the graph by using the properties of the Laplacian $L$ (since the eigenvectors of $L$ are analogous to the Fourier basis elements). The formalization is shown to preserve the localization properties at fine scales as well as other wavelets specific properties. But beyond constructing the transform, the operator-valued functions of the Laplacian are very useful to derive a powerful multiscale shape descriptor localized at different frequencies which performs very well in experiments.

For a function $f(m)$ defined on a vertex $m$ of a graph, interpreting $f(sm)$ for a scaling constant $s$, is not meaningful on its own. SGWT gets around this problem by operating in the dual domain – by taking the graph Fourier transformation. In this scenario, the spectrum of the Laplacian is analogous to the frequency domain, where scales can be defined (seen in (6) later). This provides a multi-resolution view of the signal localized at $m$. By analyzing the entire spectra at once, we can obtain a handle on which scale best characterizes the signal of interest. Indeed, for graphs, this provides a mechanism for simultaneously analyzing various local topologically-based contexts around each vertex. And for a specific scale $s$, we can now construct band-pass filters $g$ in the frequency domain which suppresses the influence of scales $s' \neq s$. When transformed back to the original domain, we directly obtain a representation of the signal for that scale. Repeating this process for multiple scales, the set of coefficients obtained for $S$ scales comprises our multiscale descriptor for that vertex.

Given a mesh with $N$ vertices, we first obtain the complete orthonormal basis $\chi_l$ and eigenvalues $\lambda_l, l \in \{0, 1, \cdots, N-1\}$ for the graph Laplacian. Using these bases, the forward and inverse graph Fourier transformation are defined using eigenvalues and eigenvectors of $L$ as,

$$\hat{f}(l) = \langle \chi_l, f \rangle = \sum_{n=1}^{N} \chi_l^*(n)f(n), \text{ and } f(n) = \sum_{l=0}^{N-1} \hat{f}(l)\chi_l(n) \tag{4}$$

Using the transforms above, we construct spectral graph wavelets by applying band-pass filters at multiple scales and localizing it with an impulse function. Since the transformed impulse function in the frequency domain is equivalent to a unit function, the wavelet $\psi$ localized at vertex $n$ is defined as,

$$\psi_{s,n}(m) = \sum_{l=0}^{N-1} g(s\lambda_l)\chi_l^*(n)\chi_l(m) \tag{5}$$

where $m$ is a vertex index on the graph. The wavelet coefficients of a given function $f(n)$ can be easily generated from the inner product of the wavelets and the given function,

$$W_f(s,n) = \langle \psi_{s,n}, f \rangle = \sum_{l=0}^{N-1} g(s\lambda_l)\hat{f}(l)\chi_l(n) \tag{6}$$

The coefficients obtained from the transformation yield the Wavelet Multiscale Descriptor (WMD) as a set of wavelet coefficients at each vertex $n$ for each scale $s$.

$$\text{WMD}_f(n) = \{W_f(s,n)|s \in S\} \tag{7}$$

In the following sections, we make use of the multi-scale descriptor for the statistical analysis of signals defined on surfaces(i.e., standard structured meshes). We will discuss shortly how many of the low-level processes in obtaining wavelet coefficients can be expressed as linear algebra primitives that can be translated on to the CUDA architecture.

# 4 Applications of Multiscale Shape Features

In this section, we present extensive experimental results demonstrating the applicability of the descriptors described above. Our core application domain is Neuroimaging. In this context, we first test if the multi-scale shape descriptors can drive significant improvements in the statistical analysis of cortical surface measurements. Then, we use these ideas to perform smoothing of cortical surface meshes *without* first projecting them onto a spherical coordinate system (the conventional approach).

## 4.1 Cortical Thickness Discrimination: Group Analysis for Alzheimer's disease (AD) studies

As we briefly discussed in Section 1, the identification of group differences between cortical surface signals is based on comparing the distribution of the signal across the two groups at each vertex. This can be done either by using the signal (cortical thickness) obtained from the segmentation directly, or by using a spherical harmonic (SPHARM) or spherical wavelet approach to first parameterize and then smooth the signal, followed by a vertex-wise $T-$test on the smoothed signal. These spherical approaches change the domain of the data from manifolds to a sphere, introducing distortion. In contrast, our multi-scale descriptor is well defined for characterizing the shape (and the signal) on the native graph domain itself. We employ hypothesis testing using the original cortical thickness and SPHARM as the two baselines for comparison when presenting our experiments below.

**Data and Pre-processing.** We used Magnetic Resonance (MR) images acquired as part of the Alzheimer's Disease Neuroimaging Initiative (ADNI). Our data included brain images from 356 participants: 160 Alzheimer's disease subjects (AD) and 196 healthy controls (CN). Details of the dataset are given in Table1.

This dataset was pre-processed using a standard image processing pipeline, and the Freesurfer algorithm [18] was used to segment the cortical surfaces, calculate the cortical thickness values, and provide vertex to vertex correspondences across brain surfaces. The data was then analyzed using our algorithm and the two baselines algorithms mentioned above.

Table 1: Demographic details and baseline cognitive status measure of the ADNI dataset

| ADNI data | | | | |
|---|---|---|---|---|
| Category | AD (mean) | AD (s.d.) | Ctrl (mean) | Ctrl (s.d.) |
| # of Subjects | 160 | - | 196 | - |
| Age | 75.53 | 7.41 | 76.09 | 5.13 |
| Gender (M/F) | 86 / 74 | - | 101 / 95 | - |
| MMSE at Baseline | 21.83 | 5.98 | 28.87 | 3.09 |
| Years of Education | 13.81 | 4.61 | 15.87 | 3.23 |

We constructed WMDs for each vertex on the template cortical surface at 6 different scales, and used Hotelling's $T^2-$test for group analysis. The same procedure was repeated using the cortical thickness measurements (from Freesurfer) and the smoothed signal obtained from SPHARM. The resulting $p$-value map was corrected for multiple comparisons over all vertices using the false discovery rate (FDR) method [2].

Fig. 2 summarizes the results of our analysis. The first row corresponds to group analysis using the original cortical thickness values (CT). Here, while we see some discriminative regions, group differences are weak and statistically significant in only a small region. The second row shows results pertaining to SPHARM, which indicate a significant improvement over the baseline, partly due to the effect of noise filtering. Finally, the bottom row in Fig. 2 shows that performing the statistical tests using our multi-scale descriptor gives substantially larger regions with much lower $p$-values. To further investigate this behavior, we repeated these experiments by making the significance level more conservative. These results (after FDR correction) are shown in Fig. 4. Again, we can directly compare CT, SPHARM and WMD for a different FDR. A very conservative FDR $q = 10^{-7}$ was used on the uncorrected $p$-values from the hypothesis test, and the $q$-values after the correction were projected back on the template mesh. Similar to Fig. 2, we see that relative to CT and SPHARM, several more regions (with substantially improved $q$-values) are recovered using the multi-scale descriptor.

To quantitatively compare the behavior above, we evaluated the uncorrected $p$-values over all vertices and sorted them in increasing order. Recall that any $p$-value below the FDR threshold is considered significant, and gives $q$-values. Fig. 3 shows the sorted $p$-values, where blue/black dotted lines are the FDR thresholds identifying significant vertices.

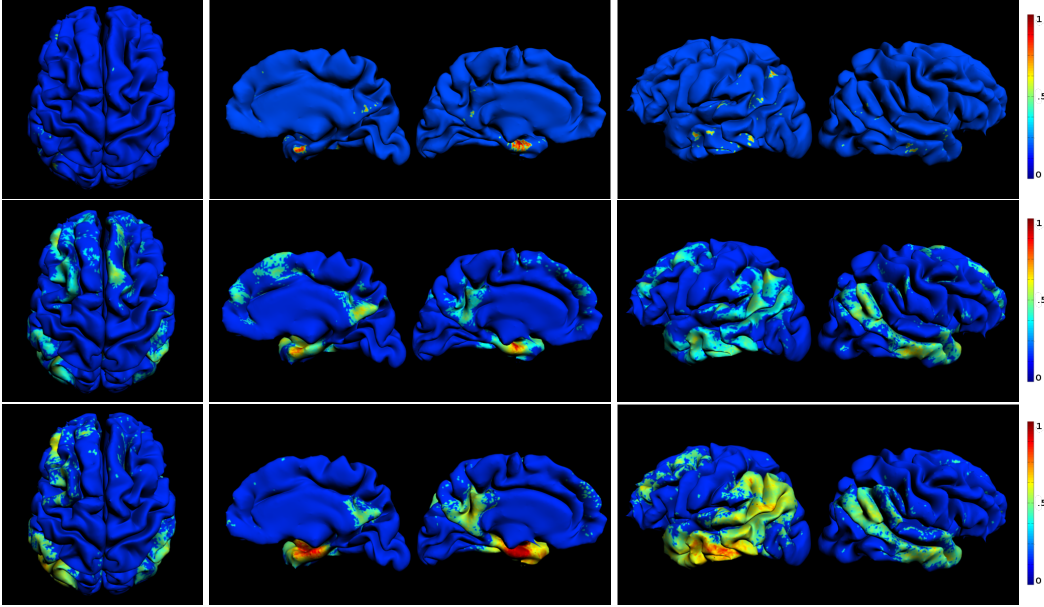

Figure 2: Normalized log scale $p$-values after FDR correction at $q = 10^{-5}$, projected back on a brain mesh and displayed. Row 1: Original cortical thickness, Row 2: SPHARM, Row 3: Wavelet Multiscale descriptor.

As seen in Figs. 2, 3 and 5, the number of significant vertices is far larger in WMD compared to CT and SPHARM. At FDR $10^{-4}$ level, there are total 6943 (CT), 28789 (SPHARM) and 40548 (WMD) vertices, showing that WMD finds 51.3% and 17.9% more discriminative vertices over CT and SPHARM methods. In Fig. 5, we can see the effect of FDR correction. With FDR set to $10^{-3}$, $10^{-5}$ and $10^{-7}$, the number of vertices that survives the correction threshold decreases to 51929, 28606 and 13226 respectively.

Finally, we evaluated the regions identified by these tests in the context of their relevance to Alzheimer's disease. We found that the identified regions are those that might be expected to be atrophic in AD. All three methods identified the anterior entorhinal cortex in the mesial temporal lobe, but at the prespecified threshold, the WMD method was more sensitive to changes in this location as well as in the posterior cingulate, precuneus, lateral parietal lobe,

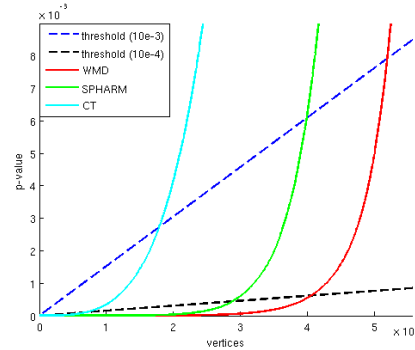

Figure 3: Sorted $p$-values from statistical analysis of sampled vertices from left hemisphere using cortical thickness (CT), SPHARM, WMD for FDR $q = 10^{-3}$ (black) and $q = 10^{-4}$ (blue).

and dorsolateral frontal lobe. These are regions that are commonly implicated in AD, and strongly tie to known results from neuroscience.

*Remarks.* When we compare two clinically different groups of brain subjects at the opposite ends of the disease spectrum (AD versus controls), the tests help identify which brain regions are severely affected. Then, if the analysis of *mild AD* versus controls reveals the same regions, we know that the new method is indeed picking up the legitimate regions. The ADNI dataset comprises of mild (and relatively younger) AD subjects, and the result from our method identifies regions which are known to be affected by AD. Our experiments suggest that for a study where group differences are expected to be weak, WMDs can facilitate identification of important variations which may be missed by the current state of the art, and can improve the statistical power of the experiment.

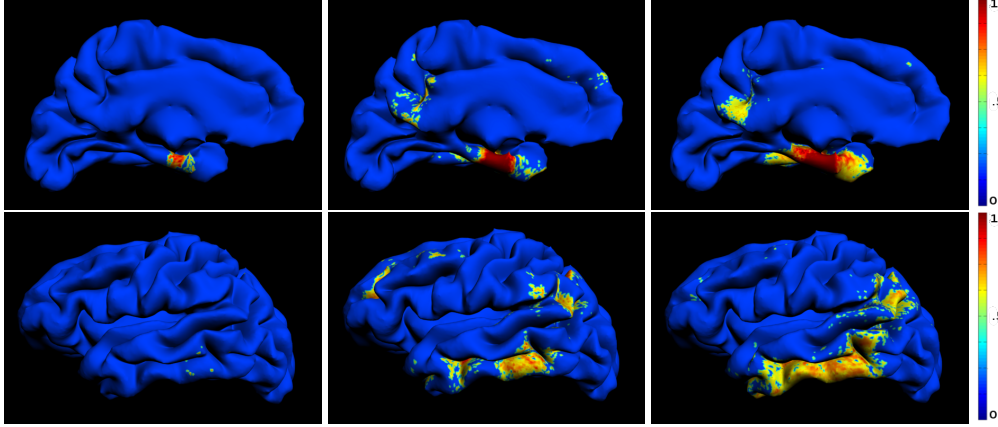

Figure 4: Normalized log scale $p$-values after FDR correction on the left hemisphere with $q = 10^{-7}$ on cortical thickness (left column) , SPHARM (middle column), WMD (right column) repectively, showing both inner and outer sides of the hemisphere.

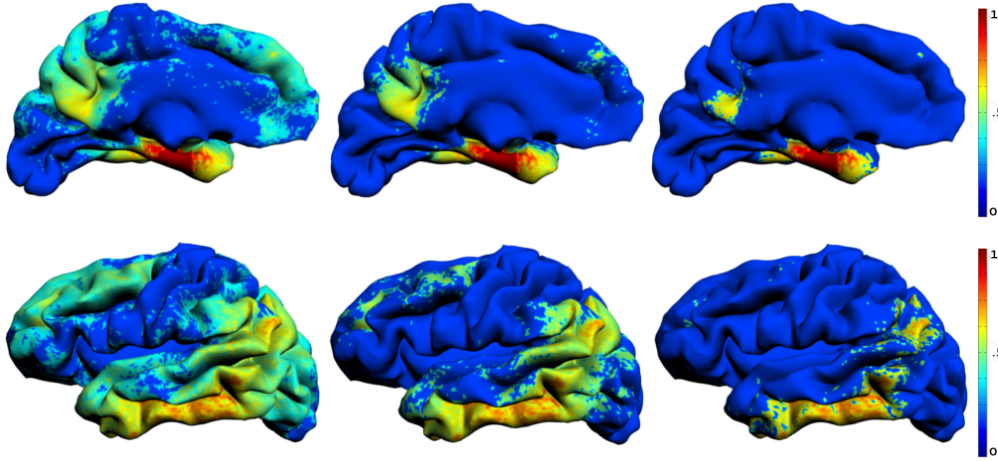

Figure 5: Normalized log scale $p$-values showing the effect of FDR correction on the template left hemisphere using WMD with FDR $q = 10^{-3}$ (left column), $q = 10^{-5}$ (middle column) and $q = 10^{-7}$ (right column) repectively, showing both inner and outer sides of the hemisphere.

## 4.2 Cortical Surface Smoothing without Sphere Mapping

Existing methods for smoothing cortical surfaces and the signal defined on it, such as spherical harmonics, explicitly represent the cortical surface as a combination of basis functions defined over regular Euclidean spaces. Such methods have been shown to be quite powerful, but invariably cause information loss due to the spherical mapping. Our goal was to evaluate whether the ideas introduced here can avoid this compromise by being able to represent (and smooth) the signal defined on any arbitrarily shaped mesh using the basis in Section 3.1 .

A small set of experiments were performed to evaluate this idea. We used wavelets of varying scales to localize the structure of the brain mesh. An inverse wavelet transformation of the resultant function provides the smooth estimate of the cortical surface at various scales. The same process can be applied to the signal defined on the surface as well. Let us rewrite (3) in terms of the graph Fourier basis, $\frac{1}{C_g} \sum_l \left( \int_0^\infty \frac{g^2(s\lambda_l)}{s} ds \right) \hat{f}(l) \chi_l(m)$ which sums over the entire scale $s$. Interestingly, in our case, the set of scales directly control the spatial smoothness of the surface. In contrast, existing methods introduce an additional smoothness parameter (e.g., $\sigma$ in case of heat kernel). Coarser spectral scales overlap less and smooth higher frequencies. At finer scale, the complete spectrum is used and recovers the original surface to high fidelity. An optimal choice of scale removes noisy high frequency variations and provide the true underlying signal. Representative examples are shown in Fig. 6 where we illustrate the process of reconstructing the surface of a brain mesh (and the cortical thickness signal) from a coarse to finer scales.

The final reconstruction of the sample brain surface from inverse transformation using five scales of wavelets and one scaling function returns total error of 2.5855 on $x$ coordinate, 2.2407 in $y$ coordinate and 2.4594 in $z$ coordinate repectively over entire $136228$ vertices. The combined error of all three coordinates per vertex is $5.346 \times 10^{-5}$, which is small. Qualitatively, we found that the results compare favorably with [6, 24] but does not need a transformation to a spherical coordinate system.

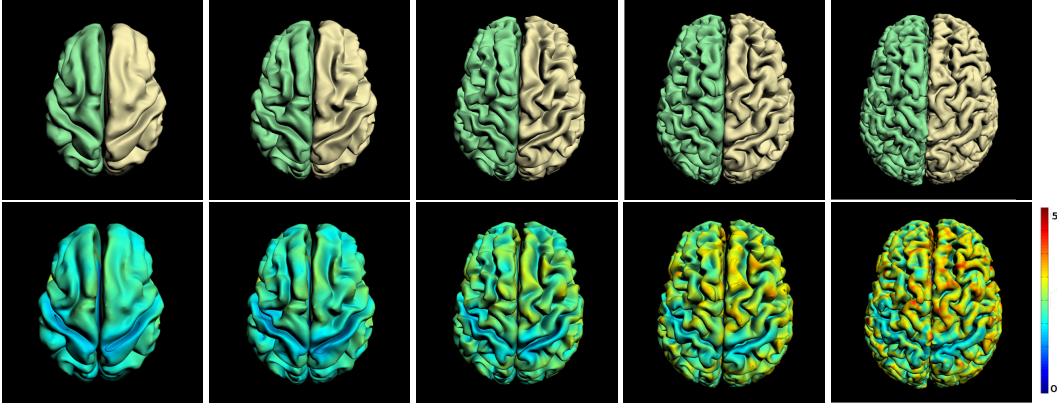

Figure 6: Structural smoothing on a brain mesh. Top row: Structural smoothing from coarse to finer scales, Bottom row: Smoothed cortical thickness displayed on the surface.

**Implementation.** Processing large surface meshes with $\sim 200000$ vertices is computationally intensive. A key bottleneck is the diagonalization of the Laplacian, which can be avoided by a clever use of a Chebyshev polynomial approximation method, as suggested by [11]. It turns out that this procedure basically consists of $n$ iterative sparse matrix-vector multiplications and scalar-vector multiplications, where $n$ is the degree of the polynomial.

With some manipulations (details in the code release), the processes above translate nicely on to the GPU architecture. Using the `cusparse` and `cublas` libraries, we derived a specialized procedure for computing the wavelet transform, which makes heavy use of commodity graphics-card hardware. Fig. 7 provides a comparison of our results to the serial MATLAB implementation and code using the commercial Jacket toolbox, for processing one brain with 166367 vertices over 6 wavelet scales as a function of polynomial degree. We see that a dataset can be processed in less than 10 seconds (even with high polynomial order) using our implementation.

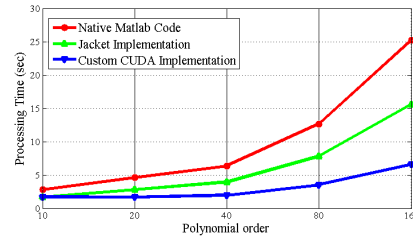

Figure 7: Running times to process a single brain dataset using native MATLAB code, Jacket, and our own implementation

## 5 Conclusions

We showed that shape descriptors based on multi-scale representations of surface based signals are a powerful tool for performing multivariate analysis of such data at various resolutions. Using a large and well characterized neuroimaging dataset, we showed how the framework improves statistical power in hypothesis testing of cortical thickness signals. We expect that in many cases, this form of analysis can detect group differences where traditional methods fail. This is the primary experimental result of the paper. We also demonstrated how the idea is applicable to cortical surface smoothing and yield competitive results without a spherical coordinate transformation. The implementation will be publicly distributed as a supplement to our paper.

**Acknowledgments**

This research was supported by funding from NIH R01AG040396, NIH R01AG021155, NSF RI 1116584, the Wisconsin Partnership Proposal, UW ADRC, and UW ICTR (1UL1RR025011). The authors are grateful to Lopa Mukherjee for much help in improving the presentation of this paper.

# References

[1] M. Belkin and P. Niyogi. Laplacian Eigenmaps for dimensionality reduction and data representation. *Neural Computation*, 15(6):pp. 1373–1396, 2003.

[2] Y. Benjamini and Y. Hochberg. Controlling the false discovery rate: A practical and powerful approach to multiple testing. *Journal of the Royal Statistical Society*, 57(1):pp. 289–300, 1995.

[3] R. Brown, N. Colter, and J. Corsellis. Postmortem evidence of structural brain changes in Schizophrenia differences in brain weight, temporal horn area, and parahippocampal gyrus compared with affective disorder. *Arch Gen Psychiatry*, 43:36–42, 1986.

[4] R. Cabin and R. Mitchell. To Bonferroni or not to Bonferroni: when and how are the questions. *Bulletin of the Ecological Society of America*, 81(3):246–248, 2000.

[5] H. Cheng, Z. Gimbutas, P. G. Martinsson, and V. Rokhlin. On the compression of low rank matrices. *SIAM J. Sci. Comput.*, 26(4):1389–1404, 2005.

[6] M. Chung, K. Dalton, S. Li, et al. Weighted Fourier series representation and its application to quantifying the amount of gray matter. *Med. Imaging, IEEE Trans. on*, 26(4):566 –581, 2007.

[7] M. Chung, K. Worsley, S. Robbins, et al. Deformation-based surface morphometry applied to gray matter deformation. *NeuroImage*, 18(2):198 – 213, 2003.

[8] R. Coifman and M. Maggioni. Diffusion wavelets. *Applied and Computational Harmonic Analysis*, 21(1):53 – 94, 2006.

[9] S. DeKosky and S. Scheff. Synapse loss in frontal cortex biopsies in Alzheimer's disease: Correlation with cognitive severity. *Annals of Neurology*, 27(5):457–464, 1990.

[10] A. Gelb. The resolution of the Gibbs phenomenon for spherical harmonics. *Mathematics of Computation*, 66:699–717, 1997.

[11] D. Hammond, P. Vandergheynst, and R. Gribonval. Wavelets on graphs via spectral graph theory. *Applied and Computational Harmonic Analysis*, 30(2):129 – 150, 2011.

[12] S. Mallat. A theory for multiresolution signal decomposition: the wavelet representation. *Pattern Analysis and Machine Intelligence, IEEE Trans. on*, 11(7):674 –693, 1989.

[13] T. Merkley, E. Bigler, E. Wilde, et al. Short communication: Diffuse changes in cortical thickness in pediatric moderate-to-severe traumatic brain injury. *Journal of Neurotrauma*, 25(11):1343–1345, 2008.

[14] K. Narr, R. Bilder, A. Toga, et al. Mapping cortical thickness and gray matter concentration in first episode Schizophrenia. *Cerebral Cortex*, 15(6):708–719, 2005.

[15] D. Pachauri, C. Hinrichs, M. Chung, et al. Topology-based kernels with application to inference problems in Alzheimer's disease. *Medical Imaging, IEEE Transactions on*, 30(10):1760 –1770, 2011.

[16] S. Peng, J. Wuu, E. Mufson, et al. Precursor form of brain-derived neurotrophic factor and mature brain-derived neurotrophic factor are decreased in the pre-clinical stages of Alzheimer's disease. *Journal of Neurochemistry*, 93(6):1412–21, 2005.

[17] E. Reiman, R. Caselli, L. Yun, et al. Preclinical evidence of Alzheimer's disease in persons homozygous for the $\varepsilon 4$ allele for apolipoprotein e. *New England Journal of Medicine*, 334(12):752–758, 1996.

[18] M. Reuter, H. D. Rosas, and B. Fischl. Highly accurate inverse consistent registration: A robust approach. *NeuroImage*, 53(4):1181–1196, 2010.

[19] O. Rioul and M. Vetterli. Wavelets and signal processing. *Signal Processing Magazine*, 8(4):14–38, 1991.

[20] P. Shaw, D. Greenstein, J. Lerch, et al. Intellectual ability and cortical development in children and adolescents. *Nature*, 440:676–679, 2006.

[21] S.Haykin and B. V. Veen. *Signals and Systems, 2nd Edition*. Wiley, 2005.

[22] E. Sowell, P. Thompson, C. Leonard, et al. Longitudinal mapping of cortical thickness and brain growth in normal children. *The Journal of Neuroscience*, 24:8223–8231, 2004.

[23] R. Sperling, P. Aisen, L. Beckett, et al. Toward defining the preclinical stages of Alzheimers disease: Recommendations from the national institute on Aging-Alzheimer's Association workgroups on diagnostic guidelines for Alzheimer's disease. *Alzheimer's and Dementia*, 7(3):280 – 292, 2011.

[24] P. Yu, P. Grant, Y. Qi, et al. Cortical surface shape analysis based on spherical wavelets. *Med. Imaging, IEEE Trans. on*, 26(4):582 –597, 2007.

